# Combining Dimensions and Features in Similarity-Based Representations

**Daniel J. Navarro**
Department of Psychology
Ohio State University
navarro.20@osu.edu

**Michael D. Lee**
Department of Psychology
University of Adelaide
michael.lee@psychology.adelaide.edu.au

## Abstract

This paper develops a new representational model of similarity data that combines continuous dimensions with discrete features. An algorithm capable of learning these representations is described, and a Bayesian model selection approach for choosing the appropriate number of dimensions and features is developed. The approach is demonstrated on a classic data set that considers the similarities between the numbers 0 through 9.

## 1  Introduction

A central problem for cognitive science is to understand the way people mentally represent stimuli. One widely used approach for deriving representations from data is to base them on measures of stimulus similarity (see Shepard 1974). Similarity is naturally understood as a measure of the degree to which the consequences of one stimulus generalize to another, and may be measured using a number of experimental methodologies, including ratings scales, confusion probabilities, or grouping or sorting tasks. For a domain with $n$ stimuli, similarity data take the form of an $n \times n$ matrix, $\mathbf{S} = [s_{ij}]$, where $s_{ij}$ is the similarity of the $i$th and $j$th stimuli. The goal of similarity-based representation is then to find structured and interpretable descriptions of the stimuli that capture the pattern of similarities.

Modeling the similarities between stimuli requires making assumptions about both the representational structures used to describe stimuli, and the processes used to assess the similarities across these structures. The two best developed representational approaches in cognitive modeling are the 'dimensional' and 'featural' approaches (Goldstone, 1999). In the dimensional approach, stimuli are represented by continuous values along a number of dimensions, so that each stimulus corresponds to a point in a multi-dimensional space, and the similarity between two stimuli is measured according to the distance between their representative points. In the featural approach, stimuli are represented in terms of the presence or absence of a set of discrete (usually binary) features or properties, and the similarity between two stimuli is measured according to their common and distinctive features.

The dimensional and featural approaches have different strengths and weaknesses. Dimensional representations are constrained by the metric axioms, such as the tri-

angle inequality, that are violated by some empirical data. Featural representations are inefficient when representing inherently continuous aspects of the variation between stimuli. It has been argued that spatial representations are most appropriate for low-level perceptual stimuli, whereas featural representations are better suited to high-level conceptual domains (e.g., Carroll 1976, Tenenbaum 1996, Tversky 1977). In general, though, stimuli convey both perceptual and conceptual information. As Carroll (1976) concludes: "Since what is going on inside the head is likely to be complex, and is equally likely to have both discrete and continuous aspects, I believe the models we pursue must also be complex, and have both discrete and continuous components" (p. 462).

This paper develops a new model of similarity that combines dimensions with features in the obvious way, allowing a stimulus to take continuous values on a number of dimensions, as well as potentially having a number of discrete features. We describe an algorithm capable of learning these representations from similarity data, and develop a Bayesian model selection approach for choosing the appropriate number of dimensions and features. Finally, we demonstrate the approach on a classic data set that considers the similarities between the numbers 0 through 9.

## 2 Dimensional, Featural and Combined Representations

### 2.1 Dimensional Representation

In a dimensional representation, the $i$th stimulus is represented by a point $\mathbf{p}_i = (p_{i1}, \ldots, p_{iv})$ in a $v$-dimensional coordinate space. The *dis*similarity between the $i$th and $j$th stimuli is then usually modeled as the distance between their points according to one of the family of Minkowskian metrics

$$\hat{d}_{ij} = \left( \sum_{k=1}^{v} |p_{ik} - p_{jk}|^r \right)^{\frac{1}{r}} + c, \tag{1}$$

where $c$ is a non-negative constant. Dimensional representations can be learned using a variety of multidimensional scaling algorithms (e.g., Cox & Cox, 1994), which have placed particular emphasis on the $r = 1$ (City-Block) and $r = 2$ (Euclidean) cases because of their relationship, respectively, to so-called 'separable' and 'integral' stimulus dimensions (Garner 1974). Pairs of separable dimensions are those, like shape and size, that can be attended to separately. Integral dimensions, in contrast, are those rarer cases like hue and saturation that are not easily separated.

### 2.2 Featural Representation

In a featural representation, the $i$th stimulus is represented by a vector of $m$ binary variables $\mathbf{f}_i = (f_{i1}, \ldots, f_{im})$, where $f_{ik} = 1$ if the $i$th stimulus possesses the $k$th feature, and $f_{ik} = 0$ if it does not. Each feature is also usually associated with a positive weight, $w_k$, denoting its importance or salience. No constraints are placed on the way features may be assigned to stimuli. Rather than requiring features partition stimuli, as in many clustering methods, or that features nest within one another, as in many tree-fitting methods, the flexible nature of human mental representation demands that features are allowed to overlap in arbitrary ways.

Although a number of models have been proposed for measuring the similarity between featurally represented stimuli (Navarro & Lee, 2002), the most widely used is the Contrast Model (Tversky, 1977). The Contrast Model assumes the similarity

between two stimuli increases according to the weights of the (common) features they share, decreases according to the weights of the (distinctive) features that one has but the other does not, and these common and distinctive sources of information are themselves weighted in arriving at a final similarity value. Particular emphasis (e.g., Shepard & Arabie, 1979; Tenenbaum, 1996) has been given to the special case of the Contrast Model where only common features are used, and feature weights are additive, so that the similarity of the $i$th and $j$th stimuli is given by

$$\hat{s}_{ij} = \sum_{k=1}^{m} w_k f_{ik} f_{jk} + c. \tag{2}$$

Although learning common feature representations is a difficult combinatorial optimization problem, several successful additive clustering algorithms have been developed (e.g., Lee, 2002; Ruml, 2001; Tenenbaum, 1996).

## 2.3 Combined Representation

The obvious generalization of dimensional and featural approaches is to represent stimuli in terms of continuous values along a set of dimensions and the presence or absence of a number of discrete features. If there are $v$ dimensions and $m$ features, the $i$th stimulus is defined by a point $\mathbf{p}_i$, a feature vector $\mathbf{f}_i$, and the feature weights $\mathbf{w} = (w_1, \ldots, w_m)$.

With this representational structure in place, we assume the similarity between the $i$th and $j$th stimuli is then simply the sum of the similarity arising from their common features (Eq. 2), minus the dissimilarity arising from their dimensional differences (Eq. 1), as follows

$$\hat{s}_{ij} = \left( \sum_{k=1}^{m} w_k f_{ik} f_{jk} \right) - \left( \sum_{k=1}^{v} |p_{ik} - p_{jk}|^r \right)^{\frac{1}{r}} + c.$$

# 3 Model Fitting and Selection

Proposing the combined representational approach immediately presents two challenges. The first *model fitting* problem is to develop a method for learning representations that fit the similarity data well using a given number of dimensions and features. The second *model selection* problem is to choose between alternative combined representations of the same data that use different numbers of features and dimensions.

Formally, we conceive of the representational model as specifying the number of dimensions and features and the nature of the distance metric, and being parameterized by the feature variables and weights, coordinate locations and the additive constant. This means a particular representation is given by $\mathbf{R}_\alpha(\theta)$ where $\alpha = (v, m, r)$ and $\theta = (\mathbf{p}_1, \ldots, \mathbf{p}_n, \mathbf{f}_1, \ldots, \mathbf{f}_n, \mathbf{w}, c)$.

Following Tenenbaum (1996), we assume that the observed similarities come from independent Gaussian distributions with means $s_{ij}$ and common variance $\sigma$. The variance corresponds to the precision of the data which, for empirical similarity data averaged across information sources (such as individual participants) is easily estimated (Lee 2001), and otherwise must be specified by assumption.

Under these assumptions, the likelihood of a similarity matrix given a particular

representation is

$$p\left(\mathbf{S} \mid \mathbf{R}_\alpha, \theta\right) = \prod_{i<j} \frac{1}{\sigma\sqrt{2\pi}} \exp\left(-\frac{1}{2\sigma^2}\left(s_{ij} - \hat{s}_{ij}\right)^2\right)$$

$$= \frac{1}{\left(\sigma\sqrt{2\pi}\right)^{n(n-1)/2}} \exp\left(-\frac{1}{2\sigma^2}\sum_{i<j}\left(s_{ij} - \hat{s}_{ij}\right)^2\right),$$

giving the log-likelihood function

$$\ln p\left(\mathbf{S} \mid \mathbf{R}_\alpha, \theta\right) = -\frac{1}{2\sigma^2}\sum_{i<j}\left(s_{ij} - \hat{s}_{ij}\right)^2 - \frac{n\left(n-1\right)}{2}\ln\left(\sigma\sqrt{2\pi}\right).$$

Within this framework, we solve the model fitting problem by finding the maximum likelihood parameter values $\theta^*$. Measures of data fit like maximum likelihood, however, are clearly not appropriate for choosing between representations with different numbers of dimensions and features, because of differences in model complexity. For this reason, we tackle the model selection problem using a Bayesian approach.

### 3.1   Fitting Algorithm

Our learning algorithm for the combined model relies on the observation (Tenenbaum, 1996) that it is relatively easy to find the maximum likelihood values of the continuous parameters—the coordinate locations, feature weights, and additive constant—given values for the discrete feature assignments.

If $\theta$ is partitioned into $\theta_C = (\mathbf{p}_1, \ldots, \mathbf{p}_n, \mathbf{w}, c)$ and a fixed $\theta_D = (\mathbf{f}_1, \ldots, \mathbf{f}_n)$, then we solve the optimization problem

$$\arg\max_{\theta_C} \ \ln p\left(\mathbf{S} \mid \mathbf{R}_\alpha, \theta_D, \theta_C\right) \qquad \text{where } \mathbf{w}, c \geq 0, \tag{3}$$

using the Levenberg-Marquardt approach (More, 1977). Since distances are preserved under translation for the Minkowskian family of metrics, we assume without loss of generality that $\mathbf{p}_1$ is the origin.

With this optimization capability in place, our learning algorithm may be described by the following five stage process:

*Step 1:* Choose a maximum number of dimensions $v_{\max}$ and features $m_{\max}$. Start with $v = 1$ and $m = 1$, making the lone feature the current feature to be optimized.

*Step 2:* Find a starting (seed) value for the current feature by considering all possibilities that have exactly one pair of stimuli with the feature, choosing the possibility with the best data-fit using Eq. 3.

*Step 3:* Consider all possible representations arising from changing the assignment of one stimulus in relation to the current feature. If any of these changes improve the fit of the representation as a whole, update the representation to be the one with the best fit. Repeat this process until no change is found that improves the representation. The current representation at this point is recorded as the best-fitting representation with $v$ dimensions and $m$ features.

*Step 4:* If there are fewer than $m_{\max}$ features, then add a new feature, make it the current feature, and return to Step 2.

*Step 5:* If there are fewer than $v_{\max}$ dimensions, then add a new dimension, reset the number of features to $m = 1$, and again make the lone feature the current feature to be optimized. Return to Step 2.

The output of this algorithm is a grid of $v_{\max} \times m_{\max}$ representations, one for each possible combination of number of dimensions and number of features.

## 3.2 Model Selection

Given representational models with different numbers of dimensions and features, the Bayesian approach is to select the one with the maximum posterior probability

$$p\left(\mathbf{R}_{\alpha} \mid \mathbf{S}\right) = \frac{p\left(\mathbf{R}_{\alpha}\right)}{p\left(\mathbf{S}\right)} \int p\left(\mathbf{S} \mid \mathbf{R}_{\alpha}, \theta\right) p\left(\theta \mid \mathbf{R}_{\alpha}\right) d\theta.$$

Since all models relate to the same similarity data, $p\left(\mathbf{S}\right)$ is a constant. If we assume that all representations are *a priori* equally likely, the posterior becomes

$$p\left(\mathbf{R}_{\alpha} \mid \mathbf{S}\right) \propto \sum_{\theta_D} \int p\left(\mathbf{S} \mid \mathbf{R}_{\alpha}, \theta\right) p\left(\theta \mid \mathbf{R}_{\alpha}\right) d\theta_C. \tag{4}$$

This Bayesian approach embodies an automatic form of Ockham's Razor, balancing data-fit against model complexity, because it considers the model at all of its parameterizations. Complicated models that use many parameters (i.e., have high parametric complexity), or parameters that interact in complicated ways (i.e., have high functional form complexity) to achieve good levels of data-fit at their optimal values will typically fit data poorly at other parameter values, and so will have smaller posteriors.

For the combined model, the posterior in Eq. 4 is not well approximated by simple measures such as the Bayesian Information (BIC: Schwarz, 1978) that have previously been applied to dimensional and featural representations (Lee & Navarro, 2002). This is because the BIC measures only parametric complexity, and treats each additional parameter as having an equal effect on model complexity. Binary feature membership parameters and continuous coordinate location parameters, however, will clearly have different effects on model complexity. In addition, because the BIC does not measure functional form complexity, it is not sensitive to the change in representational model complexity arising from different distance metrics. There are also difficulties approximating the posterior by a multivariate Gaussian with $\theta^*$ as the mode, as in the Laplacian approximation (see Kass & Raftery, 1995, p. 778), because the featural component of the combined model makes the posterior multimodal.

For these reasons, we employed Monte Carlo methods with importance sampling (e.g., Oh & Berger, 1993), in which the posterior is numerically approximated by

$$p\left(\mathbf{R}_{\alpha} \mid \mathbf{S}\right) \approx \frac{1}{N} \sum_{i=1}^{N} \frac{p\left(\mathbf{S} \mid \mathbf{R}_{\alpha}, \theta_i\right) p(\theta_i \mid \mathbf{R}_{\alpha})}{g(\theta_i \mid \mathbf{R}_{\alpha})},$$

where each of the $N$ $\theta_i$ values is independently sampled from $g(\cdot)$. In the following evaluation, we assumed that $p(\theta \mid \mathbf{R}_{\alpha})$ is uniform over $\theta$, and specified an importance distribution $g(\cdot)$ that was Gaussian over $\theta_C$ and multinomial over $\theta_D$. As the posterior may be multimodal and non-standard, $g(\cdot)$ was heavy tailed, and we sampled extensively ($N = 5 \times 10^6$) to ensure convergence.

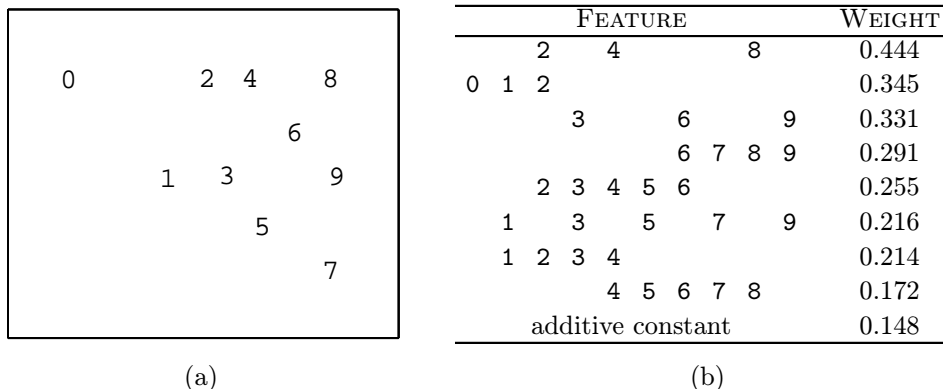

| FEATURE | | | | | | | | | | WEIGHT |
|---|---|---|---|---|---|---|---|---|---|---|
|   |   | 2 |   | 4 |   |   |   | 8 |   | 0.444 |
| 0 | 1 | 2 |   |   |   |   |   |   |   | 0.345 |
|   |   |   | 3 |   |   | 6 |   |   | 9 | 0.331 |
|   |   |   |   |   |   | 6 | 7 | 8 | 9 | 0.291 |
|   |   | 2 | 3 | 4 | 5 | 6 |   |   |   | 0.255 |
|   | 1 |   | 3 |   | 5 |   | 7 |   | 9 | 0.216 |
|   | 1 | 2 | 3 | 4 |   |   |   |   |   | 0.214 |
|   |   |   |   | 4 | 5 | 6 | 7 | 8 |   | 0.172 |
| additive constant | | | | | | | | | | 0.148 |

(a)      (b)

Figure 1: Representations of the numbers similarity data using the (a) dimensional and (b) featural approaches.

## 4  An Illustrative Example

Shepard, Kilpatric and Cunningham (1975) collected data measuring the "abstract conceptual similarity" of the numbers 0 through 9. Figure 1(a) displays a two-dimensional representation of the numbers, using the City-Block metric. This representation explains only 78.6% of the variance, and fails to capture important regularities evident in the raw data, such the fact that the number 7 is more similar to 8 than it is to 9, or that 3 is much more similar to 0 than it is to 8, and so on. Figure 1(b) shows an eight-feature representation of the numbers using the same data, as reported by Tenenbaum (1996). This representation explains 90.9% of the variance, with features corresponding to arithmetic concepts (e.g., $\{2, 4, 8\}$ and $\{3, 6, 9\}$) and to numerical magnitude (e.g., $\{1, 2, 3, 4\}$ and $\{6, 7, 8, 9\}$). We note in passing that the representations displayed in Figure 1 are also recovered when our algorithm is restricted to purely dimensional or purely featural representations.

Figure 1 suggests that the numbers data is a candidate for combined representation. Features are appropriate for representing the arithmetic concepts, but a 'magnitude' dimension seems to offer a more efficient and meaningful representation of this regularity than the five features used in Figure 1(b).

We fitted combined models with between one and three dimensions and one and eight features to the same similarity data, and calculated the log posterior for each. Because the raw data needed to estimate the precision of these averaged data are unavailable, we followed the arguments presented in Lee (2002) to make a conservative choice of $\sigma = 0.15$. The results are shown in Figure 2. All of the representations using one dimension are more likely than those using two or three dimensions. Of the one dimensional representations, the four feature version is preferred, although the likelihoods of representations with other numbers of features are close enough to warrant consideration in choosing a 'best' representation, particularly given the assumptions made about data precision.

For the sake of concreteness, however, Figure 3 describes the representation with one dimension and four features, which explains 90.0% of the variance. The one dimension almost orders the numbers according to their magnitude, with the violations being very small. The four features all capture meaningful arithmetic concepts, corresponding to "powers of two", "multiples of three", "multiples of two" (or "even

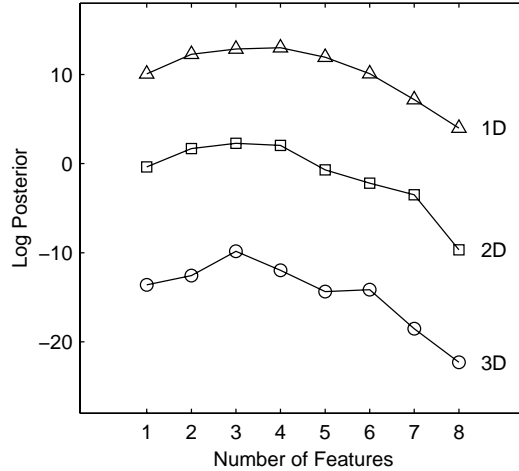

Figure 2: Log posteriors for combined representations with between one and three dimensions, and one and eight features.

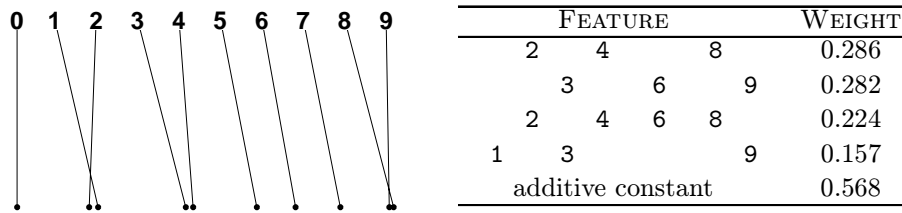

| | FEATURE | | | | WEIGHT |
|---|---|---|---|---|---|
| | 2 | 4 | | 8 | 0.286 |
| | | 3 | 6 | 9 | 0.282 |
| | 2 | 4 | 6 | 8 | 0.224 |
| 1 | 3 | | | 9 | 0.157 |
| additive constant | | | | | 0.568 |

Figure 3: Representation of the numbers similarity data using one dimension (shown on the left) and four features (shown on the right).

numbers") and "powers of three". Encouragingly, these features are close to those in Figure 1(b) that do not deal with numerical magnitude.

## 5 Conclusion

Future work will examine the use of other featural similarity models besides the purely common features approach, and will also look to develop learning algorithms that do not rely on maximum likelihood estimation, but instead consider the posterior probability of a representation. Reliable analytic approximations to the posterior will be required for this purpose.

Most importantly, however, the combined representation of a wide range of similarity data needs to be examined. Although the numbers data is a promising start, it is just a first test of the combined approach to similarity-based representation. Demonstrating the generality and usefulness of the ability to represent stimuli in terms of both dimensions and features remains a challenge for future research.

## Acknowledgments

This research was supported by Australian Research Council Grant DP0211406. We thank Tom Griffiths and two anonymous reviewers for helpful comments and discussions.

## References

[1] Carroll, J. D. (1976). Spatial, non-spatial and hybrid models for scaling. *Psychometrika, 41*, 439–463.

[2] Cox, T. F. & Cox, M. A. A. (1994). *Multidimensional Scaling.* London: Chapman and Hall.

[3] Garner, W. R. (1974).*The Processing of Information and Structure.* Potomac, MD: Erlbaum.

[4] Goldstone, R. L. (1999). Similarity. In R.A. Wilson and F.C. Keil (eds.), *MIT Encyclopedia of the Cognitive Sciences*, pp. 763–765. Cambridge, MA: MIT Press.

[5] Lee, M. D. (2001). Determining the dimensionality of multidimensional scaling representations for cognitive modeling. *Journal of Mathematical Psychology, 45*(1), 149–166.

[6] Lee, M. D. (2002). Generating additive clustering models with limited stochastic complexity. *Journal of Classification, 19*(1), 69-85.

[7] Lee, M. D. & Navarro, D. J. (2002). Extending the ALCOVE model of category learning to featural stimulus domains. *Psychonomic Bulletin & Review, 9*(1), 43-58.

[8] Kass, R. E. & Raftery, A. E. (1995). Bayes Factors. *Journal of the American Statistical Association, 90*(430), 773-795.

[9] More, J. J. (1977). The Levenberg-Marquardt algorithm: Implementation and theory. In G.A. Watson (ed.), *Lecture Notes in Mathematics, 630*, pp. 105–116. New York: Springer-Verlag.

[10] Navarro, D. J. & Lee, M. D. (2002). Commonalities and distinctions in featural stimulus representations. In: W. G. Gray, and C. D. Schunn (Eds.) *Proceedings of the 24th Annual Conference of the Cognitive Science Society*, pp. 685-690, Mahwah, NJ: Lawrence Erlbaum.

[11] Oh, M. & Berger J. O. (1993). Integration of multimodal functions by Monte Carlo importance sampling, *Journal of the American Statistical Association, 88*, 450-456.

[12] Ruml, W. (2001). Constructing distributed representations using additive clustering. In: T. G. Dietterich, S. Becker, and Z. Ghahramani (Eds.) *Advances in Neural Information Processing 14.* Cambridge, MA: MIT Press.

[13] Schwarz, G. (1978). Estimating the dimension of a model. *Annals of Statistics, 6*(2), 461–464.

[14] Shepard, R. N. (1974). Representation of structure in similarity data: Problems and prospects. *Psychometrika, 39*(4), 373–422.

[15] Shepard, R. N. & Arabie, P. (1979). Additive clustering representations of similarities as combinations of discrete overlapping properties. *Psychological Review, 86*(2), 87–123.

[16] Shepard, R. N., Kilpatric, D. W. & Cunningham, J. P. (1975). The internal representation of numbers. *Cognitive Psychology, 7*, 82–138.

[17] Tenenbaum, J. B. (1996). Learning the structure of similarity. In D. S. Touretzky, M. C. Mozer and M. E. Hasselmo (Eds.), *Advances in Neural Information Processing Systems*, pp. 3–9, Cambridge, MA: MIT Press.

[18] Tversky, A. (1977). Features of similarity. *Psychological Review, 84*(4), 327–352.